# Monotonic Networks

**Joseph Sill**
Computation and Neural Systems program
California Institute of Technology
MC 136-93, Pasadena, CA 91125
email: joe@cs.caltech.edu

## Abstract

Monotonicity is a constraint which arises in many application domains. We present a machine learning model, the monotonic network, for which monotonicity can be enforced exactly, i.e., by virtue of functional form. A straightforward method for implementing and training a monotonic network is described. Monotonic networks are proven to be universal approximators of continuous, differentiable monotonic functions. We apply monotonic networks to a real-world task in corporate bond rating prediction and compare them to other approaches.

## 1 Introduction

Several recent papers in machine learning have emphasized the importance of priors and domain-specific knowledge. In their well-known presentation of the bias-variance tradeoff (Geman and Bienenstock, 1992), Geman and Bienenstock conclude by arguing that the crucial issue in learning is the determination of the "right biases" which constrain the model in the appropriate way given the task at hand. The No-Free-Lunch theorem of Wolpert (Wolpert, 1996) shows, under the 0-1 error measure, that if all target functions are equally likely a priori, then all possible learning methods do equally well in terms of average performance over all targets. One is led to the conclusion that consistently good performance is possible only with some agreement between the modeler's biases and the true (non-flat) prior. Finally, the work of Abu-Mostafa on learning from hints (Abu-Mostafa, 1990) has shown both theoretically (Abu-Mostafa, 1993) and experimentally (Abu-Mostafa, 1995) that the use of prior knowledge can be highly beneficial to learning systems.

One piece of prior information that arises in many applications is the monotonicity constraint, which asserts that an increase in a particular input cannot result in a decrease in the output. A method was presented in (Sill and Abu-Mostafa, 1996) which enforces monotonicity approximately by adding a second term measuring

"monotonicity error" to the usual error measure. This technique was shown to yield improved error rates on real-world applications. Unfortunately, the method can be quite expensive computationally. It would be useful to have a model which obeys monotonicity exactly, i.e., by virtue of functional form.

We present here such a model, which we will refer to as a monotonic network. A monotonic network implements a piecewise-linear surface by taking maximum and minimum operations on groups of hyperplanes. Monotonicity constraints are enforced by constraining the signs of the hyperplane weights. Monotonic networks can be trained using the usual gradient-based optimization methods typically used with other models such as feedforward neural networks. Armstrong (Armstrong et. al. 1996) has developed a model called the adaptive logic network which is capable of enforcing monotonicity and appears to have some similarities to the approach presented here. The adaptive logic network, however, is available only through a commercial software package. The training algorithms are proprietary and have not been fully disclosed in academic journals. The monotonic network therefore represents (to the best of our knowledge) the first model to be presented in an academic setting which has the ability to enforce monotonicity.

Section II describes the architecture and training procedure for monotonic networks. Section III presents a proof that monotonic networks can uniformly approximate any continuous monotonic function with bounded partial derivatives to an arbitrary level of accuracy. Monotonic networks are applied to a real-world problem in bond rating prediction in Section IV. In Section V, we discuss the results and consider future directions.

## 2    Architecture and Training Procedure

A monotonic network has a feedforward, three-layer (two hidden-layer) architecture (Fig. 1). The first layer of units compute different linear combinations of the input vector. If increasing monotonicity is desired for a particular input, then all the weights connected to that input are constrained to be positive. Similarly, weights connected to an input where decreasing monotonicity is required are constrained to be negative. The first layer units are partitioned into several groups (the number of units in each group is not necessarily the same). Corresponding to each group is a second layer unit, which computes the maximum over all first-layer units within the group. The final output unit computes the minimum over all groups.

More formally, if we have $K$ groups with outputs $g_1, g_2, \ldots g_K$, and if group $k$ consists of $h_k$ hyperplanes $\mathbf{w}^{(k,1)}, \mathbf{w}^{(k,2)}, \ldots \mathbf{w}^{(k,h_k)}$, then

$$g_k(\mathbf{x}) = \max_j \mathbf{w}^{(k,j)} \cdot \mathbf{x} - t^{(k,j)}, 1 \leq j \leq h_k$$

Let $y$ be the final output of the network. Then

$$y = \min_k g_k(\mathbf{x})$$

or, for classification problems,

$$y = \sigma(\min_k g_k(\mathbf{x}))$$

where $\sigma(u) = $ e.g. $\frac{1}{1+e^{-u}}$.

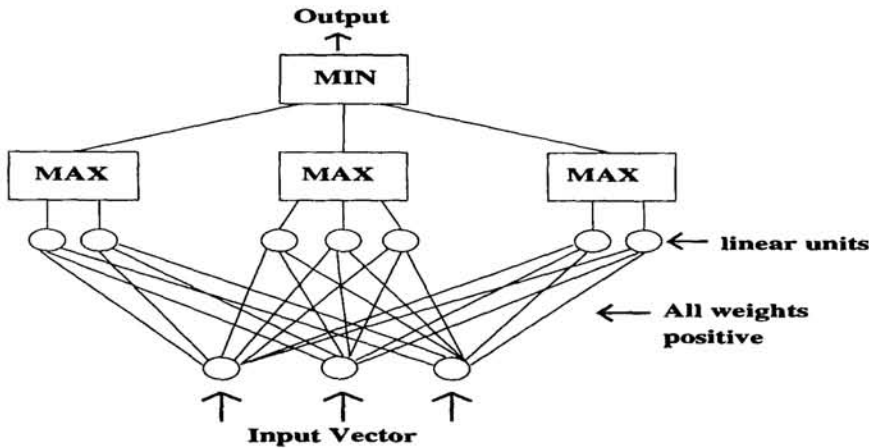

Figure 1: This monotonic network obeys increasing monotonicity in all 3 inputs because all weights in the first layer are constrained to be positive.

In the discussions which follow, it will be useful to define the term *active*. We will call a group $l$ active at $\mathbf{x}$ if

$$g_l(\mathbf{x}) = \min_k g_k(\mathbf{x})$$

, i.e., if the group determines the output of the network at that point. Similarly, we will say that a hyperplane is active at $\mathbf{x}$ if its group is active at $\mathbf{x}$ and the hyperplane is the maximum over all hyperplanes in the group.

As will be shown in the following section, the three-layer architecture allows a monotonic network to approximate any continuous, differentiable monotonic function arbitrarily well, given sufficiently many groups and sufficiently many hyperplanes within each group. The maximum operation within each group allows the network to approximate convex (positive second derivative) surfaces, while the minimum operation over groups enables the network to implement the concave (negative second derivative) areas of the target function (Figure 2).

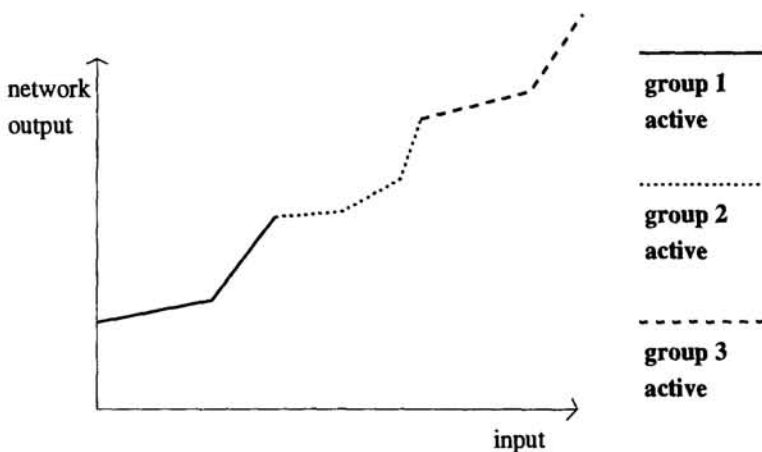

Figure 2: This surface is implemented by a monotonic network consisting of three groups. The first and third groups consist of three hyperplanes, while the second group has only two.

Monotonic networks can be trained using many of the standard gradient-based optimization techniques commonly used in machine learning. The gradient for

each hyperplane is found by computing the error over all examples for which the hyperplane is active. After the parameter update is made according to the rule of the optimization technique, each training example is reassigned to the hyperplane that is now active at that point. The set of examples for which a hyperplane is active can therefore change during the course of training.

The constraints on the signs of the weights are enforced using an exponential transformation. If increasing monotonicity is desired in input variable $i$, then $\forall j, k$ the weights corresponding to the input are represented as $w_i^{(j,k)} = e^{z_i^{(j,k)}}$. The optimization algorithm can modify $z_i^{(j,k)}$ freely during training while maintaining the constraint. If decreasing monotonicity is required, then $\forall j, k$ we take $w_i^{(j,k)} = -e^{z_i^{(j,k)}}$.

## 3   Universal Approximation Capability

In this section, we demonstrate that monotonic networks have the capacity to approximate uniformly to an arbitrary degree of accuracy any continuous, bounded, differentiable function on the unit hypercube $[0,1]^D$ which is monotonic in all variables and has bounded partial derivatives. We will say that $\mathbf{x}'$ *dominates* $\mathbf{x}$ if $\forall 1 \leq d \leq D, x_d' \geq x_d$. A function $m$ is monotonic in all variables if it satisfies the constraint that $\forall \mathbf{x}, \mathbf{x}'$, if $\mathbf{x}'$ dominates $\mathbf{x}$ then $m(\mathbf{x}') \geq m(\mathbf{x})$.

**Theorem 3.1** Let $m(\mathbf{x})$ be any continuous, bounded monotonic function with bounded partial derivatives, mapping $[0,1]^D$ to $\mathbf{R}$. Then there exists a function $m_{net}(\mathbf{x})$ which can be implemented by a monotonic network and is such that, for any $\epsilon$ and any $\mathbf{x} \in [0,1]^D$, $|m(\mathbf{x}) - m_{net}(\mathbf{x})| < \epsilon$.

**Proof:**

Let $b$ be the maximum value and $a$ be the minimum value which $m$ takes on $[0,1]^D$. Let $\alpha$ bound the magnitude of all partial first derivatives of $m$ on $[0,1]^D$. Define an equispaced grid of points on $[0,1]^D$, where $\delta = \frac{1}{n}$ is the spacing between grid points along each dimension. I.e., the grid is the set $S$ of points $(i_1 \delta, i_2 \delta, \ldots i_D \delta)$ where $1 \leq i_1 \leq n, 1 \leq i_2 \leq n, \ldots 1 \leq i_D \leq n$. Corresponding to each grid point $\mathbf{x}' = (x_1', x_2', \ldots x_D')$, assign a group consisting of $D+1$ hyperplanes. One hyperplane in the group is the constant output plane $y = m(\mathbf{x}')$. In addition, for each dimension $d$, place a hyperplane $y = \gamma(x_d - x_d') + m(\mathbf{x}')$, where $\gamma > \frac{b-a}{\delta}$. This construction ensures that the group associated with $\mathbf{x}'$ cannot be active at any point $\mathbf{x}^*$ where there exists a $d$ such that $x_d^* - x_d' > \delta$, since the group's output at such a point must be greater than $b$ and hence greater than the output of a group associated with another grid point.

Now consider any point $\mathbf{x} \in [0,1]^D$. Let $\mathbf{s}^{(1)}$ be the unique grid point in $S$ such that $\forall d, 0 \leq x_d - s_d^{(1)} < \delta$, i.e., $\mathbf{s}^{(1)}$ is the closest grid point to $\mathbf{x}$ which $\mathbf{x}$ dominates. Then we can show that $m_{net}(\mathbf{x}) \geq m(\mathbf{s}^{(1)})$. Consider an arbitrary grid point $\mathbf{s}' \neq \mathbf{s}^{(1)}$. By the monotonicity of $m$, if $\mathbf{s}'$ dominates $\mathbf{s}^{(1)}$, then $m(\mathbf{s}') \geq m(\mathbf{s}^{(1)})$, and hence, the group associated with $\mathbf{s}'$ has a constant output hyperplane $y = m(\mathbf{s}') \geq m(\mathbf{s}^{(1)})$ and therefore outputs a value $\geq m(\mathbf{s}^{(1)})$ at $\mathbf{x}$. If $\mathbf{s}'$ does not dominate $\mathbf{s}^{(1)}$, then there exists a $d$ such that $s_d^{(1)} > s_d'$. Therefore, $x_d - s_d' \geq \delta$, meaning that the output of the group associated with $\mathbf{s}'$ is at least $b \geq m(\mathbf{s}^{(1)})$. All groups have output at least as large as $m(\mathbf{s}^{(1)})$, so we have indeed shown that $m_{net}(\mathbf{x}) \geq m(\mathbf{s}^{(1)})$. Now consider the grid point $\mathbf{s}^{(2)}$ that is obtained by adding $\delta$ to each coordinate of $\mathbf{s}^{(1)}$. The group associated with $\mathbf{s}^{(2)}$ outputs $m(\mathbf{s}^{(2)})$ at $\mathbf{x}$, so $m_{net}(\mathbf{x}) \leq m(\mathbf{s}^{(2)})$. Therefore, we have $m(\mathbf{s}^{(1)}) \leq m_{net}(\mathbf{x}) \leq m(\mathbf{s}^{(2)})$. Since $\mathbf{x}$ dominates $\mathbf{s}^{(1)}$ and

is dominated by $s^{(2)}$, by monotonicity we also have $m(s^{(1)}) \leq m(x) \leq m(s^{(2)})$. $|m(x) - m_{net}(x)|$ is therefore bounded by $|m(s^{(2)}) - m(s^{(1)})|$. By Taylor's theorem for multivariate functions, we know that

$$m(s^{(2)}) - m(s^{(1)}) = \delta \sum_{d=1}^{D} \frac{\partial m(c)}{\partial x_d}$$

for some point $c$ on the line segment between $s^{(1)}$ and $s^{(2)}$. Given the assumptions made at the outset, $|m(s^{(2)}) - m(s^{(1)})|$, and hence, $|m(x) - m_{net}(x)|$ can be bounded by $d\delta\alpha$. We take $\delta < \frac{\epsilon}{d\alpha}$ to complete the proof ∎.

## 4 Experimental Results

We tested monotonic networks on a real-world problem concerning the prediction of corporate bond ratings. Rating agencies such as Standard & Poors (S & P) issue bond ratings intended to assess the level of risk of default associated with the bond. S & P ratings can range from AAA down to B- or lower.

A model which accurately predicts the S & P rating of a bond given publicly available financial information about the issuer has considerable value. Rating agencies do not rate all bonds, so an investor could use the model to assess the risk associated with a bond which S & P has not rated. The model can also be used to anticipate rating changes before they are announced by the agency.

The dataset, which was donated by a Wall Street firm, is made up of 196 examples. Each training example consists of 10 financial ratios reflecting the fundamental characteristics of the issuing firm, along with an associated rating. The meaning of the financial ratios was not disclosed by the firm for proprietary reasons. The rating labels were converted into integers ranging from 1 to 16. The task was treated as a single-output regression problem rather than a 16-class classification problem.

Monotonicity constraints suggest themselves naturally in this context. Although the meanings of the features are not revealed, it is reasonable to assume that they consist of quantities such as profitability, debt, etc. It seems intuitive that, for instance, the higher the profitability of the firm is , the stronger the firm is, and hence, the higher the bond rating should be. Monotonicity was therefore enforced in all input variables.

Three different types of models (all trained on squared error) were compared: a linear model, standard two-layer feedforward sigmoidal neural networks, and monotonic networks. The 196 examples were split into 150 training examples and 46 test examples. In order to get a statistically significant evaluation of performance, a leave-k-out procedure was implemented in which the 196 examples were split 200 different ways and each model was trained on the training set and tested on the test set for each split. The results shown are averages over the 200 splits.

Two different approaches were used with the standard neural networks. In both cases, the networks were trained for 2000 batch-mode iterations of gradient descent with momentum and an adaptive learning rate, which sufficed to allow the networks to approach minima of the training error. The first method used all 150 examples for direct training and minimized the training error as much as possible. The second technique split the 150 examples into 110 for direct training and 40 used for validation, i.e., to determine when to stop training. Specifically, the mean-squared-error on the 40 examples was monitored over the course of the 2000 iterations,

and the state of the network at the iteration where lowest validation error was obtained was taken as the final network to be tested on the test set. In both cases, the networks were initialized with small random weights. The networks had direct input-output connections in addition to hidden units in order to facilitate the implementation of the linear aspects of the target function.

The monotonic networks were trained for 1000 batch-mode iterations of gradient descent with momentum and an adaptive learning rate. The parameters of each hyperplane in the network were initialized to be the parameters of the linear model obtained from the training set, plus a small random perturbation. This procedure ensured that the network was able to find a reasonably good fit to the data. Since the meanings of the features were not known, it was not known *a priori* whether increasing or decreasing monotonicity should hold for each feature. The directions of monotonicity were determined by observing the signs of the weights of the linear model obtained from the training data.

| Model | training error | test error |
|---|---|---|
| Linear | $3.45 \pm .02$ | $4.09 \pm .06$ |
| 10-2-1 net | $1.83 \pm .01$ | $4.22 \pm .14$ |
| 10-4-1 net | $1.22 \pm .01$ | $4.86 \pm .16$ |
| 10-6-1 net | $0.87 \pm .01$ | $5.57 \pm .20$ |
| 10-8-1 net | $0.65 \pm .01$ | $5.56 \pm .16$ |

Table 1: Performance of linear model and standard networks on bond rating problem

The results support the hypothesis of a monotonic (or at least roughly monotonic) target function. As Table 1 shows, standard neural networks have sufficient flexibility to fit the training data quite accurately (n-k-1 network means a 2-layer network with n inputs, k hidden units, and 1 output). However, their excessive, non-monotonic degrees of freedom lead to overfitting, and their out-of-sample performance is even worse than that of a linear model. The use of early stopping alleviates the overfitting and enables the networks to outperform the linear model. Without the monotonicity constraint, however, standard neural networks still do not perform as well as the monotonic networks. The results seem to be quite robust with respect to the choice of number of hidden units for the standard networks and number and size of groups for the monotonic networks.

| Model | training error | test error |
|---|---|---|
| 10-2-1 net | $2.46 \pm .04$ | $3.83 \pm .09$ |
| 10-4-1 net | $2.19 \pm .05$ | $3.82 \pm .08$ |
| 10-6-1 net | $2.14 \pm .05$ | $3.77 \pm .07$ |
| 10-8-1 net | $2.13 \pm .06$ | $3.86 \pm .09$ |

Table 2: Performance of standard networks using early stopping on bond rating problem

## 5   Conclusion

We presented a model, the monotonic network, in which monotonicity constraints can be enforced exactly, without adding a second term to the usual objective function. A straightforward method for implementing and training such models was

| Model | training error | test error |
|---|---|---|
| 2 groups, 2 planes per group | $2.78 \pm .05$ | $3.71 \pm .07$ |
| 3 groups, 3 planes per group | $2.64 \pm .04$ | $3.56 \pm .06$ |
| 4 groups, 4 planes per group | $2.50 \pm .04$ | $3.48 \pm .06$ |
| 5 groups, 5 planes per group | $2.44 \pm .03$ | $3.43 \pm .06$ |

Table 3: Performance of monotonic networks on bond rating problem

demonstrated, and the method was shown to outperform other methods on a real-world problem.

Several areas of research regarding monotonic networks need to be addressed in the future. One issue concerns the choice of the number of groups and number of planes in each group. In general, the usual bias-variance tradeoff that holds for other models will apply here, and the optimal number of groups and planes will be quite difficult to determine *a priori*. There may be instances where additional prior information regarding the convexity or concavity of the target function can guide the decision, however. Another interesting observation is that a monotonic network could also be implemented by reversing the maximum and minimum operations, i.e., by taking the maximum over groups where each group outputs the minimum over all of its hyperplanes. It will be worthwhile to try to understand when one approach or the other is most appropriate.

## Acknowledgments

The author is very grateful to Yaser Abu-Mostafa for considerable guidance. I also thank John Moody for supplying the data. Amir Atiya, Eric Bax, Zehra Cataltepe, Malik Magdon-Ismail, Alexander Nicholson, and Xubo Song supplied many useful comments.

## References

[1] S. Geman and E. Bienenstock (1992). Neural Networks and the Bias-Variance Dilemma. *Neural Computation 4*, pp 1-58.

[2] D. Wolpert (1996). The Lack of A Priori Distinctions Between Learning Algorithms. *Neural Computation 8*, pp 1341-1390.

[3] Y. Abu-Mostafa (1990). Learning from Hints in Neural Networks *Journal of Complexity* 6, 192-198.

[4] Y. Abu-Mostafa (1993) Hints and the VC Dimension *Neural Computation* 4, 278-288

[5] Y. Abu-Mostafa (1995) Financial Market Applications of Learning from Hints *Neural Networks in the Capital Markets*, A. Refenes, ed., 221-232. Wiley, London, UK.

[6] J. Sill and Y. Abu-Mostafa (1996) Monotonicity Hints. To appear in it Advances in Neural Information Processing Systems 9.

[7] W.W. Armstrong, C. Chu, M. M. Thomas (1996) Feasibility of using Adaptive Logic Networks to Predict Compressor Unit Failure *Applications of Neural Networks in Environment, Energy, and Health*, Chapter 12. P. Keller, S. Hashem, L. Kangas, R. Kouzes, eds, World Scientific Publishing Company, Ltd., London.